# Efficient Offline Communication Policies for Factored Multiagent POMDPs

**João V. Messias**
Institute for Systems and Robotics
Instituto Superior Técnico
Lisbon, Portugal
jmessias@isr.ist.utl.pt

**Matthijs T.J. Spaan**
Delft University of Technology
Delft, The Netherlands
m.t.j.spaan@tudelft.nl

**Pedro U. Lima**
Institute for Systems and Robotics
Instituto Superior Técnico
Lisbon, Portugal
pal@isr.ist.utl.pt

## Abstract

Factored Decentralized Partially Observable Markov Decision Processes (Dec-POMDPs) form a powerful framework for multiagent planning under uncertainty, but optimal solutions require a rigid history-based policy representation. In this paper we allow inter-agent communication which turns the problem in a centralized Multiagent POMDP (MPOMDP). We map belief distributions over state factors to an agent's local actions by exploiting structure in the joint MPOMDP policy. The key point is that when sparse dependencies between the agents' decisions exist, often the belief over its local state factors is sufficient for an agent to unequivocally identify the optimal action, and communication can be avoided. We formalize these notions by casting the problem into convex optimization form, and present experimental results illustrating the savings in communication that we can obtain.

## 1   Introduction

Intelligent decision making in real-world scenarios requires an agent to take into account its limitations in sensing and actuation. These limitations lead to uncertainty about the state of environment, as well as how the environment will respond to performing a certain action. When multiple agents interact and cooperate in the same environment, the optimal decision-making problem is particularly challenging. For an agent in isolation, planning under uncertainty has been studied using decision-theoretic models like Partially Observable Markov Decision Processes (POMDPs) [4]. Our focus is on multiagent techniques, building on the factored Multiagent POMDP model. In this paper, we propose a novel method that exploits sparse dependencies in such a model in order to reduce the amount of inter-agent communication.

The major source of intractability for optimal Dec-POMDP solvers is that they typically reason over all possible histories of observations other agents can receive. In this work, we consider factored Dec-POMDPs in which communication between agents is possible, which has already been explored for non-factored models [10, 11, 15, 13] as well as for factored Dec-MDPs [12]. When agents share their observations at each time step, the decentralized problem reduces to a centralized one, known as a Multiagent POMDP (MPOMDP) [10]. In this work, we develop individual policies which map beliefs over state factors to actions or communication decisions.

Maintaining an exact, factorized belief state is typically not possible in cooperative problems. While bounded approximations are possible for probabilistic inference [2], these results do not carry over directly to decision-making settings (but see [5]). Intuitively, even a small difference in belief can lead to a different action being taken. However, when sparse dependencies between the agents' decisions exist, often the belief over its local state factors is sufficient for an agent to identify the action that it should take, and communication can be avoided. We formalize these notions as convex optimization problems, extracting those situations in which communication is superfluous. We present experimental results showing the savings in communication that we can obtain, and the overall impact on decision quality.

The rest of the paper is organized as follows. First, Section 2 presents the necessary background material. Section 3 presents the formalization of our method to associate belief points over state factors to actions. Next, Section 4 illustrates the concepts with experimental results, and Section 5 provides conclusions and discusses future work.

## 2 Background

In this section we provide background on factored Dec-POMDPs and Multiagent POMDPs.

A *factored* Dec-POMDP is defined as the following tuple [8]:

$\mathcal{D} = \{1, ..., n\}$ is the set of agents. $\mathcal{D}_i$ will be used to refer to agent $i$;

$\mathcal{S} = \times_i \mathcal{X}_i, i = 1, \ldots, n_f$ is the state space, decomposable into $n_f$ factors $\mathcal{X}_i \in \{1, ..., m_i\}$ which lie inside a finite range of integer values. $\mathcal{X} = \{\mathcal{X}_1, \ldots, \mathcal{X}_{n_f}\}$ is the set of all state factors;

$\mathcal{A} = \times_i \mathcal{A}_i, i = 1, ..., n$ is the joint action space. At each step, every agent $i$ takes an individual action $a_i \in \mathcal{A}_i$, resulting in the *joint* action $\mathbf{a} = \langle a_1, ..., a_n \rangle \in \mathcal{A}$;

$\mathcal{O} = \times_i \mathcal{O}_i, i = 1, ..., n$ is the space of joint observations $\mathbf{o} = \langle o_1, ..., o_n \rangle$, where $o_i \in \mathcal{O}_i$ are the individual observations. An agent receives only its own observation;

$T : \mathcal{S} \times \mathcal{S} \times \mathcal{A} \to [0, 1]$ specifies the transition probabilities $\Pr(s'|s, \mathbf{a})$;

$O : \mathcal{O} \times \mathcal{S} \times \mathcal{A} \to [0, 1]$ specifies the joint observation probabilities $\Pr(\mathbf{o}|s', \mathbf{a})$;

$R : \mathcal{S} \times \mathcal{A} \to \mathbb{R}$ specifies the reward for performing action $\mathbf{a} \in \mathcal{A}$ in state $s \in \mathcal{S}$;

$b_0 \in \mathcal{B}$ is the initial state distribution. The set $\mathcal{B}$ is the space of all possible distributions over $\mathcal{S}$;

$h$ is the planning horizon.

The main advantage of factored (Dec-)POMDP models over their standard formulation lies in their more efficient representation. Existing methods for factored Dec-POMDPs can partition the decision problem across local subsets of agents, due to the possible independence between their actions and observations [8]. A natural state-space decomposition is to perform an *agent-wise* factorization, in which a state in the environment corresponds to a unique assignment over the states of individual agents. Note that this does not preclude the existence of state factors which are common to multiple agents.

The possibility of exchanging information between agents greatly influences the overall complexity of solving a Dec-POMDP. In a fully communicative Dec-POMDP, the decentralized model can be reduced to a centralized one, the so-called *Multiagent POMDP* (MPOMDP) [10]. An MPOMDP is a regular single-agent POMDP but defined over the joint models of all agents. In a Dec-POMDP, at each $t$ an agent $i$ knows only $a_i$ and $o_i$, while in an MPOMDP, it is assumed to know $\mathbf{a}$ and $\mathbf{o}$. In the latter case, inter-agent communication is necessary to share the local observations. Solving an MPOMDP is of a lower complexity class than solving a Dec-POMDP (PSPACE-Complete vs. NEXP-Complete) [1].

It is well-known that, for a given decision step $t$, the value function $V^t$ of a POMDP is a piecewise linear, convex function [4], which can be represented as

$$V^t(b^t) = \max_{\alpha \in \Gamma^t} \alpha^{\mathrm{T}} \cdot b^t \quad , \tag{1}$$

where $\Gamma^t$ is a set of vectors (traditionally referred to as $\alpha$-vectors). Every $\alpha \in \Gamma^t$ has a particular joint action $\mathbf{a}$ associated to it, which we will denote as $\phi(\alpha)$. The transpose operator is here denoted as $(\cdot)^{\mathrm{T}}$. In this work, we assume that a value function is given for the Multiagent POMDP. However,

this value function need not be optimal, nor stationary. Our techniques preserve the quality of the supplied value function, even if it is an approximation.

A *joint* belief state is a probability distribution over the set of states $\mathcal{S}$, and encodes all of the information gathered by all agents in the Dec-POMDP up to a given time $t$:

$$b^t(s) = \Pr(s^t|\mathbf{o}^{t-1}, \mathbf{a}^{t-1}, \mathbf{o}^{t-2}, \mathbf{a}^{t-2}, \ldots, \mathbf{o}^1, \mathbf{a}^1, b_0)$$
$$= \Pr(\mathcal{X}_1^t, \ldots, \mathcal{X}_{n_f}^t|\cdot) \tag{2}$$

A factored belief state is a representation of this very same joint belief as the product of $n_F$ assumed independent belief states over the state factors $\mathcal{X}_i$, which we will refer to as *belief factors*:

$$b^t = \times_{i=1}^{n_F} b_{\mathcal{F}_i}^t \tag{3}$$

Every factor $b_{\mathcal{F}_i}^t$ is defined over a subset $\mathcal{F}_i \subseteq \mathcal{X}$ of state factors, so that:

$$b^t(s) \simeq \Pr(\mathcal{F}_1^t|\cdot)\Pr(\mathcal{F}_2^t|\cdot)\cdots\Pr(\mathcal{F}_{n_F}^t|\cdot) \tag{4}$$

With $\mathcal{F}_i \cap \mathcal{F}_j = \emptyset$ , $\forall i \neq j$. A belief point over factors $\mathcal{L}$ which are locally available to the agent will be denoted $b_{\mathcal{L}}$.

The marginalization of $b$ onto $b_{\mathcal{F}}$ is:

$$b_{\mathcal{F}}^t(\mathcal{F}^t) = \Pr\left(\mathcal{F}^t|\mathbf{a}^{1,\cdots,t-1}, \mathbf{o}^{1,\cdots,t-1}\right)$$
$$= \sum_{\mathcal{X}^t \backslash \mathcal{F}^t} \Pr\left(\mathcal{X}_1^t, \mathcal{X}_2^t, \cdots, \mathcal{X}_{n_f}^t|\cdot\right) = \sum_{\mathcal{X}^t \backslash \mathcal{F}^t} b^t(s^t), \tag{5}$$

which can be viewed as a projection of $b$ onto the smaller subspace $\mathcal{B}_{\mathcal{F}}$:

$$b_{\mathcal{F}} = M_{\mathcal{F}}^{\mathcal{X}} b \tag{6}$$

where $M_{\mathcal{F}}^{\mathcal{X}}$ is a matrix where $M_{\mathcal{F}}^{\mathcal{X}}(u,v) = 1$ if the assignments to all state factors contained in state $u \in \mathcal{F}$ are the same as in state $v \in \mathcal{X}$, and 0 otherwise. This intuitively carries out the marginalization of points in $\mathcal{B}$ onto $\mathcal{B}_{\mathcal{F}}$.

## 3 Exploiting Sparse Dependencies in Multiagent POMDPs

In the implementation of Multiagent POMDPs, an important practical issue is raised: since the joint policy arising from the value function maps joint beliefs to joint actions, all agents must maintain and update the joint belief equivalently for their decisions to remain consistent. The amount of communication required to make this possible can then become problematically large. Here, we will deal with a fully-communicative team of agents, but we will be interested in minimizing the necessary amount of communication. Even if agents can communicate with each other freely, they might not need to always do so in order to act independently, or even cooperatively.

The problem of when and what to communicate has been studied before for Dec-MDPs [12], where factors can be directly observed with no associated uncertainty, by reasoning over the possible local alternative actions to a particular assignment of observable state features. For MPOMDPs, this had been approximated at runtime, but implied keeping track and reasoning over a rapidly-growing number of possible joint belief points [11].

We will describe a method to map a belief factor (or several factors) directly to a local action, or to a communication decision, when applicable. Our approach is the first to exploit, offline, the structure of the value function itself in order to identify regions of belief space where an agent may act independently. This raises the possibility of developing more flexible forms for joint policies which can be efficiently decoupled whenever this is advantageous in terms of communication. Furthermore, since our method runs offline, it is not mutually exclusive with online communication-reduction techniques: it can be used as a basis for further computations at runtime, thereby increasing their efficiency.

## 3.1 Decision-making with factored beliefs

Note that, as fully described in [2], the factorization (4) typically results in an approximation of the true joint belief, since it is seldom possible to decouple the dynamics of a MDP into strictly independent subprocesses. The dependencies between factors, induced by the transition and observation model of the joint process, quickly develop correlations when the horizon of the decision problem is increased, even if these dependencies are sparse. Still, it was proven in [2] that, if some of these dependencies are broken, the resulting error (measured as the KL-divergence) of the factored belief state, with respect to the true joint belief, is bounded. Unfortunately, even a small error in the belief state can lead to different actions being selected, which may significantly affect the decision quality of the multiagent team in some settings [5, 9]. However, in rapidly-mixing processes (i.e., models with transition functions which quickly propagate uncertainty), the overall negative effect of using this approximation is minimized.

Each belief factor's dynamics can be described using a two-stage Dynamic Bayesian Network (DBN). For an agent to maintain, at each time step, a set of belief factors, it must have access to the state factors contained in a particular time slice of the respective DBNs. This can be accomplished either through direct observation, when possible, or by requesting this information from other agents. In the latter case, it may be necessary to perform additional communication in order to keep belief factors consistent. The amount of data to be communicated in this case, as well as its frequency, depends largely on the factorization scheme which is selected for a particular problem. We will not be here concerned with the problem of obtaining a suitable partition scheme of the joint belief onto its factors. Such a partitioning is typically simple to identify for multi-agent teams which exhibit sparsity of interaction. Instead we will focus on the amount of communication which is necessary for the joint decision-making of the multi-agent team.

## 3.2 Formal model

We will hereafter focus on the value function, and its associated quantities, at a given decision step $t$, and, for simplicity, we shall omit this dependency. However, we restate that the value function does not need to be stationary – for a finite-horizon problem, the following methods can simply be applied for every $t = 1, \ldots, h$.

### 3.2.1 Value Bounds Over Local Belief Space

Recall that, for a given $\alpha$-vector, $V_\alpha(b) = \alpha \cdot b$ represents the expected reward for selecting the action associated with $\alpha$. Ideally, if this quantity could be mapped from a local belief point $b_\mathcal{L}$, then it would be possible to select the best action for an agent based only on its local information. This is typically not possible since the projection (6) is non-invertible. However, as we will show, it is possible to obtain bounds on the achievable value of any given vector, in local belief space.

The available information regarding $V_\alpha(b)$ in local space can be expressed in the linear forms:

$$
\begin{aligned}
V_\alpha(b) &= \alpha \cdot b \\
\mathbf{1}_n^\mathrm{T} b &= 1 \\
M_\mathcal{L}^\mathcal{X} b &= b_\mathcal{L}
\end{aligned}
\tag{7}
$$

where $\mathbf{1}_n = \begin{bmatrix} 1 & 1 & \ldots & 1 \end{bmatrix}^\mathrm{T} \in \mathbb{R}^n$. Let $m$ be size of the local belief factor which contains $b_\mathcal{L}$. Reducing this system, we can associate $V_\alpha(b)$ with $b$ and $b_\mathcal{L}$, having at least $n - m$ free variables in the leading row, induced by the locally unavailable dimensions of $b$. The resulting equation can be rewritten as:

$$
V_\alpha(b) = \beta \cdot b + \gamma \cdot b_\mathcal{L} + \delta \quad,
\tag{8}
$$

with $\beta \in \mathbb{R}^n$, $\gamma \in \mathbb{R}^m$ and $\delta \in \mathbb{R}$. By maximizing (or minimizing) the terms associated with the potentially free variables, we can use this form to establish the maximum (and minimum) value that can be attained at $b_\mathcal{L}$.

**Theorem 1.** *Let* $\mathcal{I}_u = \left\{ v : M_\mathcal{L}^\mathcal{X}(u, v) = 1 \right\}$, $\overline{\beta} \in \mathbb{R}^m : \overline{\beta_i} = \max_{j \in \mathcal{I}_i} \beta_j, i = 1, \ldots, m$ *and* $\underline{\beta} \in \mathbb{R}^m : \underline{\beta_i} = \min_{j \in \mathcal{I}_i} \beta_j, i = 1, \ldots, m$. *The maximum achievable value for a local belief point,* $b_\mathcal{L}$, *according to* $\alpha$, *is:*

$$
\overline{V_\alpha}(b_\mathcal{L}) = \left( \overline{\beta} + \gamma \right) \cdot b_\mathcal{L} + \delta \quad.
\tag{9}
$$

*Analogously, the minimum achievable value is*

$$\underline{V_\alpha}(b_\mathcal{L}) = \left(\underline{\beta} + \gamma\right) \cdot b_\mathcal{L} + \delta \quad , \tag{10}$$

*Proof.* First, we shall establish that $\overline{V_\alpha}(b_\mathcal{L})$ is an upper bound on $V_\alpha(b)$. The set $\mathcal{I}_i$ contains the indexes of the elements of $b$ which marginalize onto $(b_\mathcal{L})_i$. From the definition of $\overline{\beta}$ it follows that, $\forall b \in \mathcal{B}$:

$$\sum_{j \in \mathcal{I}_i} \overline{\beta}_i b_j \geq \sum_{j \in \mathcal{I}_i} \beta_j b_j \quad , i = 1, \dots, m \quad \Leftrightarrow$$

$$\Leftrightarrow \quad \overline{\beta}_i (b_\mathcal{L})_i \geq \sum_{j \in \mathcal{I}_i} \beta_j b_j \quad , i = 1, \dots, m \quad ,$$

where we used the fact that $\sum_{j \in \mathcal{I}_i} b_j = (b_\mathcal{L})_i$. Summing over all $i$, this implies that $\overline{\beta} \cdot b_\mathcal{L} \geq \beta \cdot b$. Using (8) and (9),

$$\overline{\beta} \cdot b_\mathcal{L} + \gamma \cdot b_\mathcal{L} + \delta \geq \beta \cdot b + \gamma \cdot b_\mathcal{L} + \delta \quad \Leftrightarrow \quad \overline{V_\alpha}(b_\mathcal{L}) \geq V_\alpha(b)$$

Next, we need to show that $\exists b \in \mathcal{B} : \overline{V_\alpha}(b_\mathcal{L}) = V_\alpha(b)$. Since $\mathbf{1}_n^\mathrm{T} b = 1$ and $b_i \geq 0 \,\forall i$, $\beta \cdot b$ is a convex combination of the elements in $\beta$. Consequently,

$$\max_{b \in \mathcal{B}} \beta \cdot b = \max_{b \in \mathcal{B}} \overline{\beta} \cdot M_\mathcal{L}^\mathcal{X} b = \max_i \beta_i$$

Therefore, for $b_m = \arg\max_{b \in \mathcal{B}} \beta \cdot b$, we have that $\overline{V_\alpha}(M_\mathcal{L}^\mathcal{X} b_m) = V_\alpha(b_m)$.

The proof for the minimum achievable value $\underline{V_\alpha}(b_\mathcal{L})$ is analogous. □

By obtaining the bounds (9) and (10), we have taken a step towards identifying the correct action for an agent to take, based on the local information contained in $b_\mathcal{L}$. From their evaluation, the following remarks can be made: if $\alpha$ and $\alpha'$ are such that $\overline{V_{\alpha'}}(b_\mathcal{L}) \leq \underline{V_\alpha}(b_\mathcal{L})$, then $\alpha'$ is surely not the maximizing vector at $b$; if this property holds for all $\alpha'$ such that $(\phi(\alpha'))_i \neq (\phi(\alpha))_i$, then by following the action associated with $\alpha$, agent $i$ will accrue at least as much value as with any other vector for all possible $b$ subject to (6). That action can be safely selected without needing to communicate.

The complexity of obtaining the local value bounds for a given value function is basically that of reducing the system (7) for each vector. This is typically achieved through Gaussian Elimination, with an associated complexity of $O(n(m+2)^2)$ [3]. Note that the dominant term corresponds to the size of the local belief factor, which is usually exponentially smaller than $n$. This is repeated for all vectors, and if pruning is then done over the resulting set (the respective cost is $O(|\Gamma|^2)$), the total complexity is $O(|\Gamma|n(m+2)^2 + |\Gamma|^2)$. The pruning process used here is the same as what is typically done by POMDP solvers [14].

### 3.2.2 Dealing With Locally Ambiguous Actions

The definition of the value bounds (9) and (10) only allows an agent to act in atypical situations in which an action is clearly dominant in terms of expected value. However, this is often not the case, particularly when considering a large decision horizon, since the present effects of any given action on the overall expected reward are typically not pronounced enough for these considerations to be practical. In a situation where multiple value bounds are conflicting (i.e. $\overline{V_\alpha}(b_\mathcal{L}) > \underline{V_{\alpha'}}(b_\mathcal{L})$ and $\underline{V_\alpha}(b_\mathcal{L}) < \overline{V_{\alpha'}}(b_\mathcal{L})$), an agent is forced to further reason about which of those actions is best.

In order to tackle this problem, let us assume that two actions $a$ and $a'$ have conflicting bounds at $b_\mathcal{L}$. Given $\Gamma^a = \{\alpha \in \Gamma : (\phi(\alpha))_i = a\}$ and similarly defined $\Gamma^{a'}$, we will define the matrices $A = [\Gamma_i^a]_{k \times n}, \quad i = 1, \dots, |\Gamma^a|$ and $A' = [\Gamma_i^{a'}]_{k' \times n}, \quad i = 1, \dots, |\Gamma^{a'}|$. Then, the vectors $\mathbf{v} = Ab$ and $\mathbf{v}' = A'b$ (in $\mathbb{R}^k$ and $\mathbb{R}^{k'}$ respectively) contain all possible values attainable at $b$ through the vectors in $\Gamma^a$ and $\Gamma^{a'}$. Naturally, we will be interested in the maximum of these values for each action. In particular, we want to determine if $\max_i \mathbf{v}_i$ is greater than $\max_j \mathbf{v}'_j$ for all possible $b$ such that $b_\mathcal{L} = M_\mathcal{L}^\mathcal{X} b$. If this is the case, then $a$ should be selected as the best action, since it is guaranteed to provide a higher value at $b_\mathcal{L}$ than $a'$.

The problem of determining the minimum value of $\mathbf{v} - \mathbf{v}'$ at $b_{\mathcal{L}}$ can be expressed as the following set of Linear Programs (LPs) [6]. Note that $x \succeq y$ is here assumed to mean that $x_i \geq y_i \; \forall i$:

$$\forall i = 1, \ldots, |\Gamma^{a'}| \quad \text{maximize} \quad \Gamma_i^{a'} b - s$$
$$\text{subject to} \quad Ab \preceq \mathbf{1}_k s \quad b \succeq \mathbf{0}_n \tag{11}$$
$$M_{\mathcal{L}}^{\mathcal{X}} b = b_{\mathcal{L}} \quad \mathbf{1}_n^T b = 1$$

If the solution $b_{opt}$ to each of these LPs is such that $\max_i (Ab_{opt})_i \geq \max_j (A'b_{opt})_j$, then action $a$ can be safely selected based on $b_{\mathcal{L}}$. If this is not the case for any of the solutions, then it is not possible to map the agent's best action solely through $b_{\mathcal{L}}$. In order to disambiguate every possible action, this optimization needs to be carried out for all conflicting pairs of actions. However, a less computationally expensive alternative is to approximate the optimization (11) by a single LP (refer to [6] for more details):

$$\text{maximize} \quad \mathbf{1}_{k'}^T \xi$$
$$\text{subject to} \quad Ab \preceq \mathbf{1}_k s \quad b \succeq \mathbf{0}_n \quad M_{\mathcal{L}}^{\mathcal{X}} b = b_{\mathcal{L}} \tag{12}$$
$$A'b = \mathbf{1}_{k'} s + \xi \quad \mathbf{1}_n^T b = 1$$

### 3.2.3 Mapping Local Belief Points to Communication Decisions

For an environment with only two belief factors, the method described so far could already incorporate an explicit communication policy: given the local belief $b_{\mathcal{L}}$ of an agent, if it is possible to unequivocally identify any action as being maximal, then that action can be safely executed without any loss of expected value. Otherwise, the remaining belief factor should be requested from other agents, in order to reconstruct $b$ through (4), and map that agent's action through the joint policy. However, in most scenarios, it is not sufficient to know whether or not to communicate: equally important are the issues of what to communicate, and with whom.

Let us consider the general problem with $n_F$ belief factors contained in the set $\mathcal{F}$. In this case there are $2^{|\mathcal{F}|-1}$ combinations of non-local factors which the agent can request. Our goal is to identify one such combination which contains enough information to disambiguate the agent's actions. Central to this process is the ability to quickly determine, for a given set of belief factors $\mathcal{G} \subseteq \mathcal{F}$, if there are no points in $b_{\mathcal{G}}$ with non-decidable actions. The exact solution to this problem would require, in the worst case, the solution of $|\Gamma^a| \times |\Gamma^{a'}|$ LPs of the form (11) for every pair of actions with conflicting value bounds. However, a modification of the approximate LP (12) allows us to tackle this problem efficiently:

$$\text{maximize} \quad \mathbf{1}_{k'}^T \xi' + \mathbf{1}_k^T \xi$$
$$\text{subject to} \quad Ab \preceq \mathbf{1}_k s \quad A'b = \mathbf{1}_{k'} s + \xi \quad M_{\mathcal{L}}^{\mathcal{X}} b = b_{\mathcal{L}}$$
$$A'b' \preceq \mathbf{1}_{k'} s' \quad Ab' = \mathbf{1}_k s' + \xi' \quad M_{\mathcal{L}}^{\mathcal{X}} b' = b_{\mathcal{L}} \tag{13}$$
$$b \succeq \mathbf{0}_n \quad b' \succeq \mathbf{0}_n \quad M_{\mathcal{G}}^{\mathcal{X}} b = M_{\mathcal{G}}^{\mathcal{X}} b'$$

The rationale behind this formulation is that any solution to the LP, in which $\max_i \xi_i > 0$ and $\max_j \xi'_j > 0$ simultaneously, identifies two different points $b$ and $b'$ which map to the same point $b_G$ in $\mathcal{G}$, but share different maximizing actions $a'$ and $a$ respectively. This implies that, in order to select an action unambiguously from the belief over $\mathcal{G}$, no such solution may be possible.

Equipped with this result, we can now formulate a general procedure that, for a set of belief points in local space, returns the corresponding belief factors which must be communicated in order for an agent to act unambiguously. We refer to this as obtaining the *communication map* for the problem. This procedure is as follows (a more detailed version is included in [6]): we begin by computing the value bounds of $V$ over local factors $\mathcal{L}$, and sampling $N$ reachable local belief points $b_{\mathcal{L}}$; for each of these points, if the value bounds of the best action are not conflicting (see Section 3.2.1), or any conflicting bounds are resolved by LP (12), we can mark $b_{\mathcal{L}}$ as *safe*, add it to the communication map, and continue on to the next point; otherwise, using LP (13), we search for the minimum set of non-local factors $\mathcal{G}$ which resolves all conflicts; we then associate $b_{\mathcal{L}}$ with $\mathcal{G}$ and add it to the map.

During execution, an agent updates its local information $b_{\mathcal{L}}$, finds the nearest neighbor point in the communication map, and requests the corresponding factors from the other agents. The agent then selects the action which exhibits the highest maximum value bound given the resulting information.

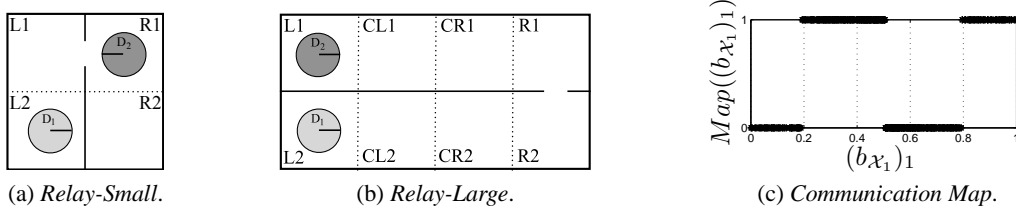

| (a) *Relay-Small*. | (b) *Relay-Large*. | (c) *Communication Map*. |

Figure 1: (a) Layout of the *Relay-Small* problem. (b) Layout of the *Relay-Large* problem. (c) Communication map for the *Relay-Small* problem.

## 4    Experiments

We now analyze the results of applying the aforementioned offline communication mapping process to three different MPOMDP environments, each with a different degrees of interdependency between agents. The first and smallest of the test problems, shown in Figure 1a, is named the *Relay-Small* problem, and is mainly used for explanatory purposes. In this world each agent is confined to a two-state area. One of the agents possesses a package which it must hand over to the other agent, through the non-traversable opening between the rooms $L1$ and $R1$. Each agent can move randomly inside its own room (a *Shuffle* action), *Exchange* the package with the other agent, or *Sense* its environment in order to find the opening. An *Exchange* is only successful if both agents are in the correct position $(L1, R1)$ and if both agents perform this action at the same time, which makes it the only available cooperative action. The fact that, in this problem, each belief factor is two-dimensional (each factor spans one of the rooms) allows us to visualize the results of our method. In Figure 2, we see that some of the agent's expected behavior is already contained in the value bounds over its local factor: if an agent is certain of being in room $R1$ (i.e. $(b_{\mathcal{X}_1})_1 = 0$), then the action with the highest-valued bound is *Shuffle*. Likewise, an *Exchange* should only be carried out when the agent is certain of being in $L1$, but it is an ambiguous action since the agent needs to be sure that its teammate can cooperate. In Figure 1c we represent the communication map which was obtained offline through the proposed algorithm. Since there are only two factors, the agent only needs to make a binary decision of whether or not to communicate for a given local belief point. The belief points considered *safe* are marked as $0$, and those associated with a communication decision are marked as $1$. In terms of quantitative results, we see that $\sim 30 - 40\%$ of communication episodes are avoided in this simple example, without a significant loss of collected reward.

Another test scenario is the OneDoor environment of [7], which is further described in [6]. In this 49-state world, two agents lie inside opposite rooms, akin to the *Relay-Small* problem, but each agent has the goal of moving to the other room. There is only one common passage between both rooms, where the agents may collide. For shorter-horizon solutions, agents may not be able to reach their goal, and they communicate so as to minimize negative reward (collisions). For the infinite-horizon case, however, typically only one of the agents communicates, while waiting for its partner to clear the passage. Note that this relationship between the problem's horizon and the amount of communication savings does not hold for all of the problems. The proposed method exploits the invariance of local policies over subsets of the joint belief space, and this may arbitrarily change with the problem's horizon.

A larger example is displayed in Figure 1b. This is an adaptation of the *Relay-Small* problem (aptly named *Relay-Large*) to a setting in which each room has four different states, and each agent may be carrying a package at a given time. Agent $D_1$ may retrieve new packages from position L1, and $D_2$

| | Relay-Small | | OneDoor | | Relay-Large | |
|---|---|---|---|---|---|---|
| h. | Full Comm. | Red. Comm. | Full Comm. | Red. Comm. | Full Comm. | Red. Comm. |
| 6 | 15.4, 100% | 14.8, 56.9% | 0.35, 100% | 0.30, 89.0% | 27.4, 100% | 25.8, 44.1% |
| 10 | 39.8, 100% | 38.7, 68.2% | 1.47, 100% | 1.38, 76.2% | -19.7, 100% | -21.6, 62,5% |
| $\infty$ | 77.5, 100% | 73.9, 46.1% | 2.31, 100% | 2.02, 61.3% | 134.0, 100% | 129.7, 58.9% |

Table 1: Results of the proposed method for various environments. For settings assuming full and reduced communication, we show empirical control quality, online communication usage.

|           | Relay-Small |      |          | OneDoor |      |          | Relay-Large |       |          |
|-----------|------|------|----------|------|------|----------|-------|-------|----------|
| h         | 6    | 10   | $\infty$ | 6    | 10   | $\infty$ | 6     | 10    | $\infty$ |
| Perseus   | 1.1  | 4.3  | 0.1      | 7.3  | 33.3 | 5.3      | 239.5 | 643.0 | 31.5     |
| Comm. Map | 5.9  | 21.4 | 7.4      | 12.4 | 57.7 | 5.9      | 368.7 | 859.5 | 138.1    |

Table 2: Running time (in seconds) of the proposed method in comparison to the Perseus point-based POMDP solver.

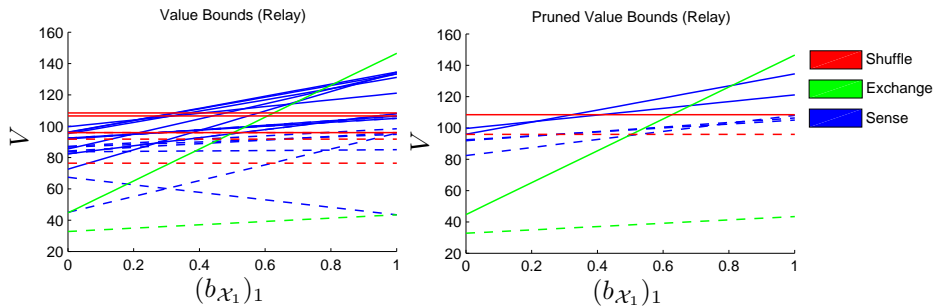

Figure 2: Value bounds for the *Relay-Small* problem. The dashed lines indicate the minimum value bounds, and the filled lines represent the maximum value bounds, for each action.

can deliver them to L2, receiving for that a positive reward. There are a total of $64$ possible states for the environment. Here, since the agents can act independently for a longer time, the communication savings are more pronounced, as shown in Table 1.

Finally, we argue that the running time of the proposed algorithm is comparable to that of general POMDP solvers for these same environments. Even though both the solver and the mapper algorithms must be executed in sequence, the results in Table 2 show that they are typically both in the same order of magnitude.

## 5   Conclusions and Future Work

Traditional multiagent planning on partially observable environments mostly deals with fully-communicative or non-communicative situations. For a more realistic scenario where communication should be used only when necessary, state-of-the-art methods are only capable of approximating the optimal policy at run-time [11, 15]. Here, we have analyzed the properties of MPOMDP models which can be exploited in order to increase the efficiency of communication between agents. We have shown that these properties hold, for various MPOMDP scenarios, and that the decision quality can be maintained while significantly reducing the amount of communication, as long as the dependencies within the model are sparse.

Although one of the main features of these techniques is that they may be applied to any given MPOMDP value function, in some situations this value function may be costly to obtain. As future work, we will investigate methods for obtaining MPOMDP value functions that are easy to partition using our techniques.

**Acknowledgments**

This work was funded in part by Fundação para a Ciência e a Tecnologia (ISR/IST pluriannual funding) through the PIDDAC Program funds and was supported by project CMU-PT/SIA/0023/2009 under the Carnegie Mellon-Portugal Program. J.M. was supported by a PhD Student Scholarship, SFRH/BD/44661/2008, from the Portuguese FCT POCTI programme. M.S. is funded by the FP7 Marie Curie Actions Individual Fellowship #275217 (FP7-PEOPLE-2010-IEF).

# References

[1] Daniel S. Bernstein, Robert Givan, Neil Immerman, and Shlomo Zilberstein. The complexity of decentralized control of Markov decision processes. *Mathematics of Operations Research*, 27(4):819–840, 2002.

[2] Xavier Boyen and Daphne Koller. Tractable inference for complex stochastic processes. In *Proc. of Uncertainty in Artificial Intelligence*, 1998.

[3] X.G. Fang and G. Havas. On the worst-case complexity of integer gaussian elimination. In *Proceedings of the 1997 international symposium on Symbolic and algebraic computation*, pages 28–31. ACM, 1997.

[4] L. P. Kaelbling, M. L. Littman, and A. R. Cassandra. Planning and acting in partially observable stochastic domains. *Artificial Intelligence*, 101:99–134, 1998.

[5] David A. McAllester and Satinder Singh. Approximate planning for factored POMDPs using belief state simplification. In *Proc. of Uncertainty in Artificial Intelligence*, 1999.

[6] J.V. Messias, M.T.J. Spaan, and P. U. Lima. Supplementary material for "Efficient Offline Communication Policies for Factored Multiagent POMDPs". ISR/IST, 2011.

[7] Frans A. Oliehoek, Matthijs T. J. Spaan, and Nikos Vlassis. Dec-POMDPs with delayed communication. In *Multi-agent Sequential Decision Making in Uncertain Domains*, 2007. Workshop at AAMAS07.

[8] Frans A. Oliehoek, Matthijs T. J. Spaan, Shimon Whiteson, and Nikos Vlassis. Exploiting locality of interaction in factored Dec-POMDPs. In *Proc. of Int. Conference on Autonomous Agents and Multi Agent Systems*, 2008.

[9] P. Poupart and C. Boutilier. Value-directed belief state approximation for POMDPs. In *Proc. of Uncertainty in Artificial Intelligence*, volume 130, 2000.

[10] David V. Pynadath and Milind Tambe. The communicative multiagent team decision problem: Analyzing teamwork theories and models. *Journal of Artificial Intelligence Research*, 16:389–423, 2002.

[11] M. Roth, R. Simmons, and M. Veloso. Decentralized communication strategies for coordinated multi-agent policies. In *Multi-Robot Systems: From Swarms to Intelligent Automata*, volume IV. Kluwer Academic Publishers, 2005.

[12] Maayan Roth, Reid Simmons, and Manuela Veloso. Exploiting factored representations for decentralized execution in multi-agent teams. In *Proc. of Int. Conference on Autonomous Agents and Multi Agent Systems*, 2007.

[13] Matthijs T. J. Spaan, Frans A. Oliehoek, and Nikos Vlassis. Multiagent planning under uncertainty with stochastic communication delays. In *Proc. of Int. Conf. on Automated Planning and Scheduling*, pages 338–345, 2008.

[14] Chelsea C. White. Partially observed Markov decision processes: a survey. *Annals of Operations Research*, 32, 1991.

[15] Feng Wu, Shlomo Zilberstein, and Xiaoping Chen. Multi-agent online planning with communication. In *Int. Conf. on Automated Planning and Scheduling*, 2009.

